# Variational Inference for Bayesian Mixtures of Factor Analysers

**Zoubin Ghahramani** and **Matthew J. Beal**

Gatsby Computational Neuroscience Unit
University College London
17 Queen Square, London WC1N 3AR, England
{zoubin,m.beal}@gatsby.ucl.ac.uk

## Abstract

We present an algorithm that infers the model structure of a mixture of factor analysers using an efficient and deterministic variational approximation to full Bayesian integration over model parameters. This procedure can automatically determine the optimal number of components and the local dimensionality of each component (i.e. the number of factors in each factor analyser). Alternatively it can be used to infer posterior distributions over number of components and dimensionalities. Since all parameters are integrated out the method is not prone to overfitting. Using a stochastic procedure for adding components it is possible to perform the variational optimisation incrementally and to avoid local maxima. Results show that the method works very well in practice and correctly infers the number and dimensionality of nontrivial synthetic examples.

By importance sampling from the variational approximation we show how to obtain unbiased estimates of the true evidence, the exact predictive density, and the KL divergence between the variational posterior and the true posterior, not only in this model but for variational approximations in general.

## 1 Introduction

Factor analysis (FA) is a method for modelling correlations in multidimensional data. The model assumes that each $p$-dimensional data vector $\mathbf{y}$ was generated by first linearly transforming a $k < p$ dimensional vector of unobserved independent zero-mean unit-variance Gaussian sources, $\mathbf{x}$, and then adding a $p$-dimensional zero-mean Gaussian noise vector, $\mathbf{n}$, with diagonal covariance matrix $\Psi$: i.e. $\mathbf{y} = \Lambda\mathbf{x} + \mathbf{n}$. Integrating out $\mathbf{x}$ and $\mathbf{n}$, the marginal density of $\mathbf{y}$ is Gaussian with zero mean and covariance $\Lambda\Lambda^T + \Psi$. The matrix $\Lambda$ is known as the factor loading matrix. Given data with a sample covariance matrix $\Sigma$, factor analysis finds the $\Lambda$ and $\Psi$ that optimally fit $\Sigma$ in the maximum likelihood sense. Since $k < p$, a single factor analyser can be seen as a reduced parametrisation of a full-covariance Gaussian.[1]

A *mixture* of factor analysers (MFA) models the density for **y** as a weighted average of factor analyser densities

$$P(\mathbf{y}|\Lambda,\Psi,\pi) = \sum_{s=1}^{S} P(s|\pi)P(\mathbf{y}|s,\Lambda^s,\Psi),\tag{1}$$

where $\pi$ is the vector of mixing proportions, $s$ is a discrete indicator variable, and $\Lambda^s$ is the factor loading matrix for factor analyser $s$ which includes a mean vector for **y**.

By exploiting the factor analysis parameterisation of covariance matrices, a mixture of factor analysers can be used to fit a mixture of Gaussians to correlated high dimensional data without requiring $O(p^2)$ parameters or undesirable compromises such as axis-aligned covariance matrices. In an MFA each Gaussian cluster has intrinsic dimensionality $k$ (or $k_s$ if the dimensions are allowed to vary across clusters). Consequently, the mixture of factor analysers simultaneously addresses the problems of clustering and local dimensionality reduction. When $\Psi$ is a multiple of the identity the model becomes a mixture of probabilistic PCAs. Tractable maximum likelihood procedure for fitting MFA and MPCA models can be derived from the Expectation Maximisation algorithm [4, 11].

The maximum likelihood (ML) approach to MFA can easily get caught in local maxima.[2] Ueda et al. [12] provide an effective deterministic procedure for avoiding local maxima by considering splitting a factor analyser in one part of space and merging two in a another part. But splits and merges have to be considered simultaneously because the number of factor analysers has to stay the same since adding a factor analyser is always expected to increase the training likelihood.

A fundamental problem with maximum likelihood approaches is that they fail to take into account model complexity (i.e. the cost of coding the model parameters). So more complex models are not penalised, which leads to overfitting and the inability to determine the best model size and structure (or distributions thereof) without resorting to costly cross-validation procedures. Bayesian approaches overcome these problems by treating the parameters $\theta$ as unknown random variables and averaging over the ensemble of models they define:

$$P(Y) = \int d\theta\, P(Y|\theta)P(\theta).\tag{2}$$

$P(Y)$ is the *evidence* for a data set $Y = \{\mathbf{y}^1,\ldots,\mathbf{y}^N\}$. Integrating out parameters penalises models with more degrees of freedom since these models can *a priori* model a larger range of data sets. All information inferred from the data about the parameters is captured by the posterior distribution $P(\theta|Y)$ rather than the ML point estimate $\hat{\theta}$.[3]

While Bayesian theory deals with the problems of overfitting and model selection/averaging, in practice it is often computationally and analytically intractable to perform the required integrals. For Gaussian mixture models *Markov chain Monte Carlo* (MCMC) methods have been developed to approximate these integrals by sampling [8, 7]. The main criticism of MCMC methods is that they are slow and

[2]Technically, the log likelihood is not bounded above if no constraints are put on the determinant of the component covariances. So the real ML objective for MFA is to find the highest finite local maximum of the likelihood.

[3]We sometimes use $\theta$ to refer to the parameters and sometimes to all the unknown quantities (parameters and hidden variables). Formally the only difference between the two is that the number of hidden variables grows with $N$, whereas the number of parameters usually does not.

it is usually difficult to assess convergence. Furthermore, the posterior density over parameters is stored as a set of samples, which can be inefficient.

Another approach to Bayesian integration for Gaussian mixtures [9] is the *Laplace approximation* which makes a local Gaussian approximation around a maximum *a posteriori* parameter estimate. These approximations are based on large data limits and can be poor, particularly for small data sets (for which, in principle, the advantages of Bayesian integration over ML are largest). Local Gaussian approximations are also poorly suited to bounded or positive parameters such as the mixing proportions of the mixture model. Finally, it is difficult to see how this approach can be applied to online incremental changes to model structure.

In this paper we employ a third approach to Bayesian inference: *variational approximation*. We form a lower bound on the log evidence using Jensen's inequality:

$$\mathcal{L} \equiv \ln P(Y) = \ln \int d\theta \, P(Y,\theta) \geq \int d\theta \, Q(\theta) \ln \frac{P(Y,\theta)}{Q(\theta)} \equiv \mathcal{F}, \qquad (3)$$

which we seek to maximise. Maximising $\mathcal{F}$ is equivalent to minimising the KL-divergence between $Q(\theta)$ and $P(\theta|Y)$, so a tractable $Q$ can be used as an approximation to the intractable posterior. This approach draws its roots from one way of deriving mean field approximations in physics, and has been used recently for Bayesian inference [13, 5, 1].

The variational method has several advantages over MCMC and Laplace approximations. Unlike MCMC, convergence can be assessed easily by monitoring $\mathcal{F}$. The approximate posterior is encoded efficiently in $Q(\theta)$. Unlike Laplace approximations, the form of $Q$ can be tailored to each parameter (in fact the optimal form of $Q$ for each parameter falls out of the optimisation), the approximation is global, and $Q$ optimises an objective function. Variational methods are generally fast, $\mathcal{F}$ is guaranteed to increase monotonically and transparently incorporates model complexity. To our knowledge, no one has done a full Bayesian analysis of mixtures of factor analysers.

Of course, vis-a-vis MCMC, the main disadvantage of variational approximations is that they are not guaranteed to find the exact posterior in the limit. However, with a straightforward application of sampling, it is possible to take the result of the variational optimisation and use it to sample from the exact posterior and exact predictive density. This is described in section 5.

In the remainder of this paper we first describe the mixture of factor analysers in more detail (section 2). We then derive the variational approximation (section 3). We show empirically that the model can infer both the number of components and their intrinsic dimensionalities, and is not prone to overfitting (section 6). Finally, we conclude in section 7.

## 2   The Model

Starting from (1), the evidence for the Bayesian MFA is obtained by averaging the likelihood under priors for the parameters (which have their own hyperparameters):

$$
\begin{aligned}
P(Y) \;=\; & \int d\pi \, P(\pi|\alpha) \int d\nu \, P(\nu|a,b) \int d\Lambda \, P(\Lambda|\nu) \cdot \\
& \prod_{n=1}^{N} \left[ \sum_{s^n=1}^{S} P(s^n|\pi) \int d\mathbf{x}^n P(\mathbf{x}^n) P(\mathbf{y}^n|\mathbf{x}^n, s^n, \Lambda^s, \Psi) \right]. \qquad (4)
\end{aligned}
$$

Here $\{\alpha, a, b, \Psi\}$ are hyperparameters[4], $\nu$ are precision parameters (i.e. inverse variances) for the columns of $\Lambda$. The conditional independence relations between the variables in this model are shown graphically in the usual belief network representation in Figure 1.

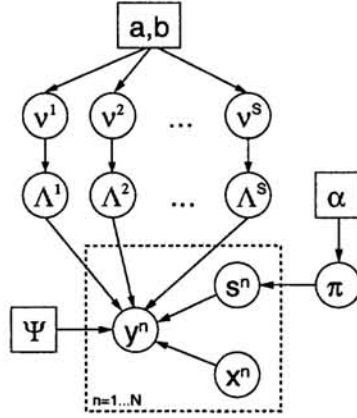

Figure 1: Generative model for variational Bayesian mixture of factor analysers. Circles denote random variables, solid rectangles denote hyperparameters, and the dashed rectangle shows the plate (i.e. repetitions) over the data.

While arbitrary choices could be made for the priors on the first line of (4), choosing priors that are conjugate to the likelihood terms on the second line of (4) greatly simplifies inference and interpretability.[5] So we choose $P(\pi|\alpha)$ to be symmetric Dirichlet, which is conjugate to the multinomial $P(s|\pi)$.

The prior for the factor loading matrix plays a key role in this model. Each component of the mixture has a Gaussian prior $P(\Lambda^s|\nu^s)$, where each element of the vector $\nu^s$ is the precision of a *column* of $\Lambda$. If one of these precisions $\nu^s_l \to \infty$, then the outgoing weights for factor $\mathbf{x}_l$ will go to zero, which allows the model to reduce the intrinsic dimensionality of $\mathbf{x}$ if the data does not warrant this added dimension. This method of intrinsic dimensionality reduction has been used by Bishop [2] for Bayesian PCA, and is closely related to MacKay and Neal's method for automatic relevance determination (ARD) for inputs to a neural network [6].

To avoid overfitting it is important to integrate out all parameters whose cardinality scales with model complexity (i.e. number of components and their dimensionalities). We therefore also integrate out the precisions using Gamma priors, $P(\nu|a, b)$.

## 3    The Variational Approximation

Applying Jensen's inequality repeatedly to the log evidence (4) we lower bound it using the following factorisation of the distribution of parameters and hidden variables: $Q(\Lambda)Q(\pi, \nu)Q(s, \mathbf{x})$. Given this factorisation several additional factorisations fall out of the conditional independencies in the model resulting in the variational objective function:

$$\mathcal{F} = \int d\pi Q(\pi) \ln \frac{P(\pi|\alpha)}{Q(\pi)} + \sum_{s=1}^{S} \int d\nu^s Q(\nu^s) \left[ \ln \frac{P(\nu^s|a,b)}{Q(\nu^s)} + \int d\Lambda^s Q(\Lambda^s) \ln \frac{P(\Lambda^s|\nu^s)}{Q(\Lambda^s)} \right]$$

$$+ \sum_{n=1}^{N} \sum_{s^n=1}^{S} Q(s^n) \left[ \int d\pi \, Q(\pi) \ln \frac{P(s^n|\pi)}{Q(s^n)} + \int d\mathbf{x}^n Q(\mathbf{x}^n|s^n) \ln \frac{P(\mathbf{x}^n)}{Q(\mathbf{x}^n|s^n)} \right.$$

$$\left. + \int d\Lambda^s Q(\Lambda^s) \int d\mathbf{x}^n Q(\mathbf{x}^n|s^n) \ln P(\mathbf{y}^n|\mathbf{x}^n, s^n, \Lambda^s, \Psi) \right] \quad (5)$$

The variational posteriors $Q(\cdot)$, as given in the Appendix, are derived by performing a free-form extremisation of $\mathcal{F}$ w.r.t. $Q$. It is not difficult to show that these extrema are indeed maxima of $\mathcal{F}$. The optimal posteriors $Q$ are of the same conjugate forms as the priors. The model hyperparameters which govern the priors can be estimated in the same fashion (see the Appendix).

## 4 Birth and Death

When optimising $\mathcal{F}$, occasionally one finds that for some $s$: $\sum_n Q(s^n) = 0$. These zero responsibility components are the result of there being insufficient support from the local data to overcome the dimensional complexity prior on the factor loading matrices. So components of the mixture die of natural causes when they are no longer needed. Removing these redundant components increases $\mathcal{F}$.

Component birth does not happen spontaneously, so we introduce a heuristic. Whenever $\mathcal{F}$ has stabilised we pick a parent-component stochastically with probability proportional to $e^{-\beta \mathcal{F}_s}$ and attempt to split it into two; $\mathcal{F}_s$ is the $s$-specific contribution to $\mathcal{F}$ with the last bracketed term in (5) normalised by $\sum_n Q(s^n)$. This works better than both cycling through components and picking them at random as it concentrates attempted births on components that are faring poorly. The parameter distributions of the two Gaussians created from the split are initialised by partitioning the responsibilities for the data, $Q(s^n)$, along a direction sampled from the parent's distribution. This usually causes $\mathcal{F}$ to decrease, so by monitoring the future progress of $\mathcal{F}$ we can reject this attempted birth if $\mathcal{F}$ does not recover.

Although it is perfectly possible to start the model with many components and let them die, it is computationally more efficient to start with one component and allow it to spawn more when necessary.

## 5 Exact Predictive Density, True Evidence, and KL

By importance sampling from the variational approximation we can obtain unbiased estimates of three important quantities: the exact predictive density, the true log evidence $\mathcal{L}$, and the KL divergence between the variational posterior and the true posterior. Letting $\theta = \{\Lambda, \pi\}$, we sample $\theta_i \sim Q(\theta)$. Each such sample is an instance of a mixture of factor analysers with predictive density given by (1). We weight these predictive densities by the importance weights $w_i = P(\theta_i, Y)/Q(\theta_i)$, which are easy to evaluate. This results in a *mixture* of mixtures of factor analysers, and will converge to the exact predictive density, $P(\mathbf{y}|Y)$, as long as $Q(\theta) > 0$ wherever $P(\theta|Y) > 0$. The true log evidence can be similarly estimated by $\mathcal{L} = \ln\langle w \rangle$, where $\langle \cdot \rangle$ denotes averaging over the importance samples. Finally, the KL divergence is given by: $\mathrm{KL}(Q(\theta)\|P(\theta|Y)) = \ln\langle w \rangle - \langle \ln w \rangle$.

This procedure has three significant properties. First, the same importance weights can be used to estimate all three quantities. Second, while importance sampling can work very poorly in high dimensions for ad hoc proposal distributions, here the variational optimisation is used in a principled manner to pick $Q$ to be a good approximation to $P$ and therefore hopefully a good proposal distribution. Third, this procedure can be applied to any variational approximation. A detailed exposition can be found in [3].

## 6 Results

**Experiment 1: Discovering the number of components.** We tested the model on synthetic data generated from a mixture of 18 Gaussians with 50 points per cluster (Figure 2, top left). The variational algorithm has little difficulty finding the correct number of components and the birth heuristics are successful at avoiding local maxima. After finding the 18 Gaussians repeated splits are attempted and rejected. Finding a distribution over number of components using $\mathcal{F}$ is also simple.

**Experiment 2: The shrinking spiral.** We used the dataset of 800 data points from a shrinking spiral from [12] as another test of how well the algorithm could

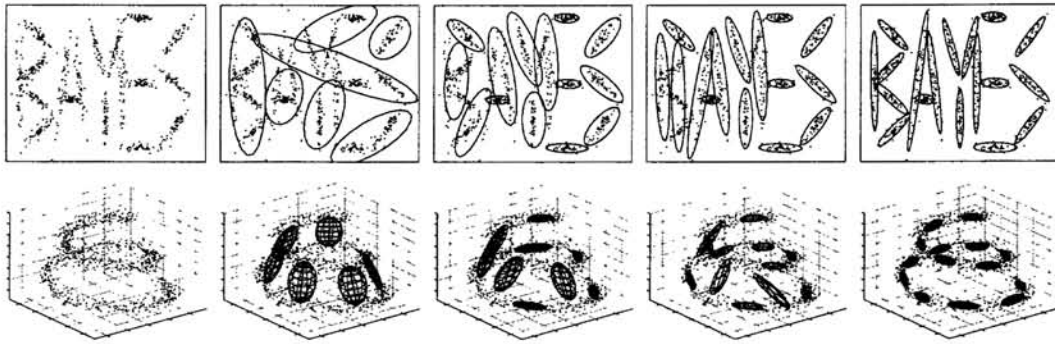

Figure 2: **(top)** Exp 1: The frames from left to right are the data, and the 2 S.D. Gaussian ellipses after 7, 14, 16 and 22 accepted births. **(bottom)** Exp 2: Shrinking spiral data and 1 S.D. Gaussian ellipses after 6, 9, 12, and 17 accepted births. Note that the number of Gaussians increases from left to right.

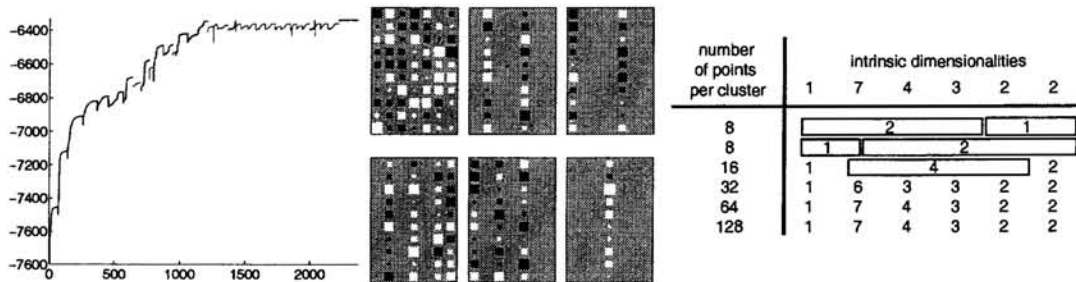

| number of points per cluster | intrinsic dimensionalities | | | | | |
|---|---|---|---|---|---|---|
| | 1 | 7 | 4 | 3 | 2 | 2 |
| 8 | | 2 | | | 1 | |
| 8 | 1 | | 2 | | | |
| 16 | 1 | 4 | | | | 2 |
| 32 | 1 | 6 | 3 | 3 | 2 | 2 |
| 64 | 1 | 7 | 4 | 3 | 2 | 2 |
| 128 | 1 | 7 | 4 | 3 | 2 | 2 |

Figure 3: **(left)** Exp 2: $\mathcal{F}$ as function of iteration for the spiral problem on a typical run. Drops in $\mathcal{F}$ constitute component births. Thick lines are accepted attempts, thin lines are rejected attempts. **(middle)** Exp 3: Means of the factor loading matrices. These results are analogous to those given by Bishop [2] for Bayesian PCA. **(right)** Exp 3: Table with learned number of Gaussians and dimensionalities as training set size increases. Boxes represent model components that capture several of the clusters.

escape local maxima and how robust it was to initial conditions (Figure 2, bottom). Again local maxima did not pose a problem and the algorithm always found between 12-14 Gaussians regardless of whether it was initialised with 0 or 200. These runs took about 3-4 minutes on a 500MHz Alpha EV6 processor. A plot of $\mathcal{F}$ shows that most of the compute time is spent on accepted moves (Figure 3, left).

**Experiment 3: Discovering the local dimensionalities.**  We generated a synthetic data set of 300 data points in each of 6 Gaussians with intrinsic dimensionalities (7 4 3 2 2 1) embedded in 10 dimensions. The variational Bayesian approach correctly inferred both the number of Gaussians and their intrinsic dimensionalities (Figure 3, middle). We varied the number of data points and found that as expected with fewer points the data could not provide evidence for as many components and intrinsic dimensions (Figure 3, right).

## 7   Discussion

Search over model structures for MFAs is computationally intractable if each factor analyser is allowed to have different intrinsic dimensionalities. In this paper we have shown that the variational Bayesian approach can be used to efficiently infer this model structure while avoiding overfitting and other deficiencies of ML approaches. One attraction of our variational method, which can be exploited in other models, is that once a factorisation of $Q$ is assumed all inference is automatic and exact. We can also use $\mathcal{F}$ to get a distribution over structures if desired. Finally we derive

a generally applicable importance sampler that gives us unbiased estimates of the true evidence, the exact predictive density, and the KL divergence between the variational posterior and the true posterior.

Encouraged by the results on synthetic data, we have applied the Bayesian mixture of factor analysers to a real-world unsupervised digit classification problem. We will report the results of these experiments in a separate article.

## Appendix: Optimal $Q$ Distributions and Hyperparameters

$$Q(\mathbf{x}^n|s^n) \sim \mathcal{N}(\overline{\mathbf{x}}^{n,s}, \Sigma^s) \quad Q(\Lambda_q^s) \sim \mathcal{N}(\overline{\Lambda}_q^s, \Sigma^{q,s}) \quad Q(\boldsymbol{\nu}_l^s) \sim \mathcal{G}(a_l^s, b_l^s) \quad Q(\boldsymbol{\pi}) \sim \mathcal{D}(\omega \mathbf{u})$$

$$\ln Q(s^n) = [\psi(\omega u_s) - \psi(\omega)] + \frac{1}{2}\ln|\Sigma^s| + \langle \ln P(\mathbf{y}^n|\mathbf{x}^n, s^n, \Lambda^s, \Psi)\rangle + c$$

$$\overline{\mathbf{x}}^{n,s} = \Sigma^s \overline{\Lambda}^{s\top} \Psi^{-1} \mathbf{y}^n, \quad \overline{\Lambda}_q^s = \left[\Psi^{-1}\sum_{n=1}^{N} Q(s^n)\mathbf{y}^n\overline{\mathbf{x}}^{n,s\top}\Sigma^{q,s}\right]_q, \quad a_l^s = a + \frac{p}{2}, \quad b_l^s = b + \frac{1}{2}\sum_{q=1}^{p}\langle\Lambda_{ql}^{s}{}^2\rangle$$

$$\Sigma^{s-1} = \langle\Lambda^{s\top}\Psi^{-1}\Lambda^s\rangle + I, \quad \Sigma^{q,s-1} = \Psi_{qq}^{-1}\sum_{n=1}^{N} Q(s^n)\langle\mathbf{x}^n\mathbf{x}^{n\top}\rangle + \text{diag}\langle\boldsymbol{\nu}^s\rangle, \quad \omega u_s = \frac{\alpha}{S} + \sum_{n=1}^{N} Q(s^n)$$

where $\{\mathcal{N}, \mathcal{G}, \mathcal{D}\}$ denote Normal, Gamma and Dirichlet distributions respectively, $\langle\cdot\rangle$ denotes expectation under the variational posterior, and $\psi(x)$ is the *digamma* function $\psi(x) \equiv \frac{\partial}{\partial x}\ln\Gamma(x)$. Note that the optimal distributions $Q(\Lambda^s)$ have block diagonal covariance structure; even though each $\Lambda^s$ is a $p \times q$ matrix, its covariance only has $O(pq^2)$ parameters. Differentiating $\mathcal{F}$ with respect to the parameters, $a$ and $b$, of the precision prior we get fixed point equations $\psi(a) = \langle\ln\boldsymbol{\nu}\rangle + \ln b$ and $b = a/\langle\boldsymbol{\nu}\rangle$. Similarly the fixed point for the parameters of the Dirichlet prior is $\psi(\alpha) - \psi(\alpha/S) + \sum[\psi(\omega u_s) - \psi(\omega)]/S = 0$.

## Footnotes

[1] Factor analysis and its relationship to principal components analysis (PCA) and mixture models is reviewed in [10].

[4]We currently do not integrate out $\Psi$, although this can also be done.

[5]Conjugate priors have the same effect as pseudo-observations.

## References

[1] H. Attias. Inferring parameters and structure of latent variable models by variational Bayes. In *Proc. 15th Conf. on Uncertainty in Artificial Intelligence*, 1999.

[2] C.M. Bishop. Variational PCA. In *Proc. Ninth Int. Conf. on Artificial Neural Networks. ICANN*, 1999.

[3] Z. Ghahramani, H. Attias, and M.J. Beal. Learning model structure. Technical Report GCNU-TR-1999-006, (in prep.) Gatsby Unit, Univ. College London, 1999.

[4] Z. Ghahramani and G.E. Hinton. The EM algorithm for mixtures of factor analyzers. Technical Report CRG-TR-96-1 [http://www.gatsby.ucl.ac.uk/~zoubin/papers/tr-96-1.ps.gz], Dept. of Comp. Sci., Univ. of Toronto, 1996.

[5] D.J.C. MacKay. Ensemble learning for hidden Markov models. Technical report, Cavendish Laboratory, University of Cambridge, 1997.

[6] R.M. Neal. Assessing relevance determination methods using DELVE. In C.M. Bishop, editor, *Neural Networks and Machine Learning*, 97–129. Springer-Verlag, 1998.

[7] C.E. Rasmussen. The infinite gaussian mixture model. In *Adv. Neur. Inf. Proc. Sys. 12*. MIT Press, 2000.

[8] S. Richardson and P.J. Green. On Bayesian analysis of mixtures with an unknown number of components. *J. Roy. Stat. Soc.-Ser. B*, 59(4):731–758, 1997.

[9] S.J. Roberts, D. Husmeier, I. Rezek, and W. Penny. Bayesian approaches to Gaussian mixture modeling. *IEEE PAMI*, 20(11):1133–1142, 1998.

[10] S. T. Roweis and Z. Ghahramani. A unifying review of linear Gaussian models. *Neural Computation*, 11(2):305–345, 1999.

[11] M.E. Tipping and C.M. Bishop. Mixtures of probabilistic principal component analyzers. *Neural Computation*, 11(2):443–482, 1999.

[12] N. Ueda, R. Nakano, Z. Ghahramani, and G.E. Hinton. SMEM algorithm for mixture models. In *Adv. Neur. Inf. Proc. Sys. 11*. MIT Press, 1999.

[13] S. Waterhouse, D.J.C. Mackay, and T. Robinson. Bayesian methods for mixtures of experts. In *Adv. Neur. Inf. Proc. Sys. 7*. MIT Press, 1995.
